# The Relevance Vector Machine

**Michael E. Tipping**
Microsoft Research
St George House, 1 Guildhall Street
Cambridge CB2 3NH, U.K.
mtipping@microsoft.com

## Abstract

The support vector machine (SVM) is a state-of-the-art technique for regression and classification, combining excellent generalisation properties with a sparse kernel representation. However, it does suffer from a number of disadvantages, notably the absence of probabilistic outputs, the requirement to estimate a trade-off parameter and the need to utilise 'Mercer' kernel functions. In this paper we introduce the *Relevance Vector Machine* (RVM), a Bayesian treatment of a generalised linear model of identical functional form to the SVM. The RVM suffers from none of the above disadvantages, and examples demonstrate that for comparable generalisation performance, the RVM requires dramatically fewer kernel functions.

## 1 Introduction

In *supervised learning* we are given a set of examples of input vectors $\{\mathbf{x}_n\}_{n=1}^N$ along with corresponding targets $\{t_n\}_{n=1}^N$, the latter of which might be real values (in *regression*) or class labels (*classification*). From this 'training' set we wish to learn a model of the dependency of the targets on the inputs with the objective of making accurate predictions of $t$ for previously unseen values of $\mathbf{x}$. In real-world data, the presence of noise (in regression) and class overlap (in classification) implies that the principal modelling challenge is to avoid 'over-fitting' of the training set.

A very successful approach to supervised learning is the *support vector machine* (SVM) [8]. It makes predictions based on a function of the form

$$y(\mathbf{x}) = \sum_{n=1}^{N} w_n K(\mathbf{x}, \mathbf{x}_n) + w_0, \tag{1}$$

where $\{w_n\}$ are the model 'weights' and $K(\cdot, \cdot)$ is a *kernel* function. The key feature of the SVM is that, in the classification case, its target function attempts to minimise the number of errors made on the training set while simultaneously maximising the 'margin' between the two classes (in the feature space implicitly defined by the kernel). This is an effective 'prior' for avoiding over-fitting, which leads to good generalisation, and which furthermore results in a *sparse* model dependent only on a subset of kernel functions: those associated with training examples $\mathbf{x}_n$ that lie either on the margin or on the 'wrong' side of it. State-of-the-art results have been reported on many tasks where SVMs have been applied.

However, the support vector methodology does exhibit significant disadvantages:

- Predictions are not *probabilistic*. In regression the SVM outputs a point estimate, and in classification, a 'hard' binary decision. Ideally, we desire to estimate the conditional distribution $p(t|\mathbf{x})$ in order to capture uncertainty in our prediction. In regression this may take the form of 'error-bars', but it is particularly crucial in classification where posterior probabilities of class membership are necessary to adapt to varying class priors and asymmetric misclassification costs.

- Although relatively sparse, SVMs make liberal use of kernel functions, the requisite number of which grows steeply with the size of the training set.

- It is necessary to estimate the error/margin trade-off parameter '$C$' (and in regression, the insensitivity parameter '$\epsilon$' too). This generally entails a cross-validation procedure, which is wasteful both of data and computation.

- The kernel function $K(\cdot, \cdot)$ must satisfy Mercer's condition.

In this paper, we introduce the 'relevance vector machine' (RVM), a probabilistic sparse kernel model identical in functional form to the SVM. Here we adopt a Bayesian approach to learning, where we introduce a prior over the weights governed by a set of hyperparameters, one associated with each weight, whose most probable values are iteratively estimated from the data. Sparsity is achieved because in practice we find that the posterior distributions of many of the weights are sharply peaked around zero. Furthermore, unlike the support vector classifier, the non-zero weights in the RVM are *not* associated with examples close to the decision boundary, but rather appear to represent 'prototypical' examples of classes. We term these examples 'relevance' vectors, in deference to the principle of *automatic relevance determination* (ARD) which motivates the presented approach [4, 6].

The most compelling feature of the RVM is that, while capable of generalisation performance comparable to an equivalent SVM, it typically utilises dramatically fewer kernel functions. Furthermore, the RVM suffers from none of the other limitations of the SVM outlined above.

In the next section, we introduce the Bayesian model, initially for regression, and define the procedure for obtaining hyperparameter values, and thus weights. In Section 3, we give brief examples of application of the RVM in the regression case, before developing the theory for the classification case in Section 4. Examples of RVM classification are then given in Section 5, concluding with a discussion.

## 2 Relevance Vector Regression

Given a dataset of input-target pairs $\{\mathbf{x}_n, t_n\}_{n=1}^N$, we follow the standard formulation and assume $p(t|\mathbf{x})$ is Gaussian $\mathcal{N}(t|y(\mathbf{x}), \sigma^2)$. The mean of this distribution for a given $\mathbf{x}$ is modelled by $y(\mathbf{x})$ as defined in (1) for the SVM. The likelihood of the dataset can then be written as

$$p(\mathbf{t}|\mathbf{w}, \sigma^2) = (2\pi\sigma^2)^{-N/2} \exp\left\{-\frac{1}{2\sigma^2}\|\mathbf{t} - \mathbf{\Phi w}\|^2\right\}, \tag{2}$$

where $\mathbf{t} = (t_1 \ldots t_N)$, $\mathbf{w} = (w_0 \ldots w_N)$ and $\mathbf{\Phi}$ is the $N \times (N+1)$ 'design' matrix with $\mathbf{\Phi}_{nm} = K(\mathbf{x}_n, \mathbf{x}_{m-1})$ and $\mathbf{\Phi}_{n1} = 1$. Maximum-likelihood estimation of $\mathbf{w}$ and $\sigma^2$ from (2) will generally lead to severe overfitting, so we encode a preference for smoother functions by defining an ARD Gaussian prior [4, 6] over the weights:

$$p(\mathbf{w}|\boldsymbol{\alpha}) = \prod_{i=0}^N \mathcal{N}(w_i|0, \alpha_i^{-1}), \tag{3}$$

with $\boldsymbol{\alpha}$ a vector of $N+1$ hyperparameters. This introduction of an individual hyperparameter for every weight is the key feature of the model, and is ultimately responsible for its sparsity properties. The posterior over the weights is then obtained from Bayes' rule:

$$p(\mathbf{w}|\mathbf{t},\boldsymbol{\alpha},\sigma^2) = (2\pi)^{-(N+1)/2}|\boldsymbol{\Sigma}|^{-1/2}\exp\left\{-\frac{1}{2}(\mathbf{w}-\boldsymbol{\mu})^{\mathsf{T}}\boldsymbol{\Sigma}^{-1}(\mathbf{w}-\boldsymbol{\mu})\right\}, \quad (4)$$

with

$$\boldsymbol{\Sigma} = (\boldsymbol{\Phi}^{\mathsf{T}}\mathbf{B}\boldsymbol{\Phi}+\mathbf{A})^{-1}, \quad (5)$$

$$\boldsymbol{\mu} = \boldsymbol{\Sigma}\boldsymbol{\Phi}^{\mathsf{T}}\mathbf{B}\mathbf{t}, \quad (6)$$

where we have defined $\mathbf{A} = \mathrm{diag}(\alpha_0,\alpha_1,\ldots,\alpha_N)$ and $\mathbf{B} = \sigma^{-2}\mathbf{I}_N$. Note that $\sigma^2$ is also treated as a hyperparameter, which may be estimated from the data.

By integrating out the weights, we obtain the *marginal likelihood*, or *evidence* [2], for the hyperparameters:

$$p(\mathbf{t}|\boldsymbol{\alpha},\sigma^2) = (2\pi)^{-N/2}|\mathbf{B}^{-1}+\boldsymbol{\Phi}\mathbf{A}^{-1}\boldsymbol{\Phi}^{\mathsf{T}}|^{-1/2}\exp\left\{-\frac{1}{2}\mathbf{t}^{\mathsf{T}}(\mathbf{B}^{-1}+\boldsymbol{\Phi}\mathbf{A}^{-1}\boldsymbol{\Phi}^{\mathsf{T}})^{-1}\mathbf{t}\right\}. \quad (7)$$

For ideal Bayesian inference, we should define hyperpriors over $\boldsymbol{\alpha}$ and $\sigma^2$, and integrate out the hyperparameters too. However, such marginalisation cannot be performed in closed-form here, so we adopt a pragmatic procedure, based on that of MacKay [2], and *optimise* the marginal likelihood (7) with respect to $\boldsymbol{\alpha}$ and $\sigma^2$, which is essentially the *type II maximum likelihood* method [1]. This is equivalent to finding the maximum of $p(\boldsymbol{\alpha},\sigma^2|\mathbf{t})$, assuming a uniform (and thus improper) hyperprior. We then make predictions, based on (4), using these maximising values.

## 2.1 Optimising the hyperparameters

Values of $\boldsymbol{\alpha}$ and $\sigma^2$ which maximise (7) cannot be obtained in closed form, and we consider two alternative formulae for iterative re-estimation of $\boldsymbol{\alpha}$. First, by considering the weights as 'hidden' variables, an EM approach gives:

$$\alpha_i^{\mathrm{new}} = \frac{1}{\langle w_i^2\rangle_{p(\mathbf{w}|\mathbf{t},\boldsymbol{\alpha},\sigma^2)}} = \frac{1}{\Sigma_{ii}+\mu_i^2}. \quad (8)$$

Second, direct differentiation of (7) and rearranging gives:

$$\alpha_i^{\mathrm{new}} = \frac{\gamma_i}{\mu_i^2}, \quad (9)$$

where we have defined the quantities $\gamma_i = 1 - \alpha_i\Sigma_{ii}$, which can be interpreted as a measure of how 'well-determined' each parameter $w_i$ is by the data [2]. Generally, this latter update was observed to exhibit faster convergence.

For the noise variance, both methods lead to the same re-estimate:

$$(\sigma^2)^{\mathrm{new}} = \|\mathbf{t}-\boldsymbol{\Phi}\boldsymbol{\mu}\|^2/(N-\sum_i\gamma_i). \quad (10)$$

In practice, during re-estimation, we find that many of the $\alpha_i$ approach infinity, and from (4), $p(w_i|\mathbf{t},\boldsymbol{\alpha},\sigma^2)$ becomes infinitely peaked at zero — implying that the corresponding kernel functions can be 'pruned'. While space here precludes a detailed explanation, this occurs because there is an 'Occam' penalty to be paid for smaller values of $\alpha_i$, due to their appearance in the determinant in the marginal likelihood (7). For some $\alpha_i$, a lesser penalty can be paid by explaining the data with increased noise $\sigma^2$, in which case those $\alpha_i \to \infty$.

# 3 Examples of Relevance Vector Regression

## 3.1 Synthetic example: the 'sinc' function

The function $\text{sinc}(x) = |x|^{-1} \sin |x|$ is commonly used to illustrate support vector regression [8], where in place of the classification margin, the *ε-insensitive region* is introduced, a 'tube' of $\pm\epsilon$ around the function within which errors are not penalised. In this case, the support vectors lie on the edge of, or outside, this region. For example, using linear spline kernels and with $\epsilon = 0.01$, the approximation of $\text{sinc}(x)$ based on 100 uniformly-spaced noise-free samples in $[-10, 10]$ utilises 39 support vectors [8].

By comparison, we approximate the same function with a relevance vector model utilising the same kernel. In this case the noise variance is fixed at $0.01^2$ and $\alpha$ alone re-estimated. The approximating function is plotted in Figure 1 (left), and requires only 9 relevance vectors. The largest error is 0.0087, compared to 0.01 in the SV case. Figure 1 (right) illustrates the case where Gaussian noise of standard deviation 0.2 is added to the targets. The approximation uses 6 relevance vectors, and the noise is automatically estimated, using (10), as $\sigma = 0.189$.

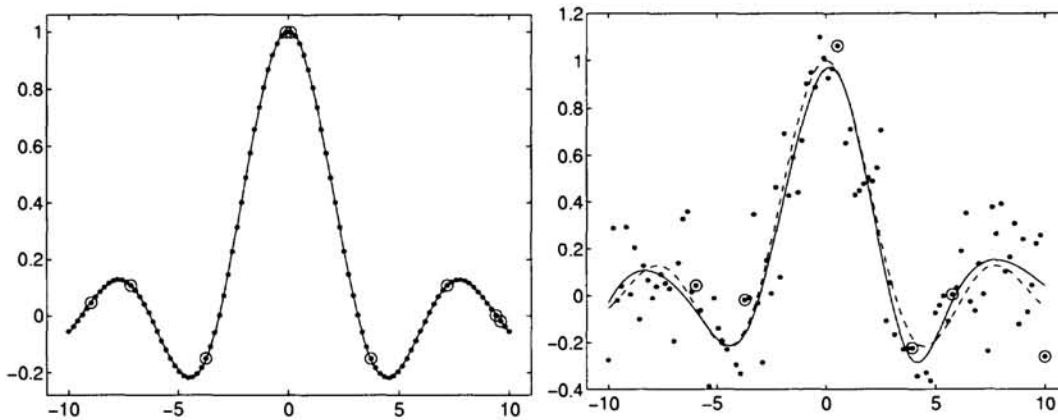

Figure 1: Relevance vector approximation to $\text{sinc}(x)$: noise-free data (left), and with added Gaussian noise of $\sigma = 0.2$ (right). The estimated functions are drawn as solid lines with relevance vectors shown circled, and in the added-noise case (right) the true function is shown dashed.

## 3.2 Some benchmarks

The table below illustrates regression performance on some popular benchmark datasets — Friedman's three synthetic functions (results averaged over 100 randomly generated training sets of size 240 with a 1000-example test set) and the 'Boston housing' dataset (averaged over 100 randomised 481/25 train/test splits). The prediction error obtained and the number of kernel functions required for both support vector regression (SVR) and relevance vector regression (RVR) are given.

| Dataset | errors SVR | errors RVR | kernels SVR | kernels RVR |
|---|---|---|---|---|
| Friedman #1 | 2.92 | 2.80 | 116.6 | 59.4 |
| Friedman #2 | 4140 | 3505 | 110.3 | 6.9 |
| Friedman #3 | 0.0202 | 0.0164 | 106.5 | 11.5 |
| Boston Housing | 8.04 | 7.46 | 142.8 | 39.0 |

## 4   Relevance Vector Classification

We now extend the relevance vector approach to the case of classification — i.e. where it is desired to predict the posterior probability of class membership given the input $\mathbf{x}$. We generalise the linear model by applying the logistic sigmoid function $\sigma(y) = 1/(1 + e^{-y})$ to $y(\mathbf{x})$ and writing the likelihood as

$$P(\mathbf{t}|\mathbf{w}) = \prod_{n=1}^{N} \sigma\{y(\mathbf{x}_n)\}^{t_n} \left[1 - \sigma\{y(\mathbf{x}_n)\}\right]^{1-t_n}. \qquad (11)$$

However, we cannot integrate out the weights to obtain the marginal likelihood analytically, and so utilise an iterative procedure based on that of MacKay [3]:

1. For the current, fixed, values of $\boldsymbol{\alpha}$ we find the most probable weights $\mathbf{w}_{\mathrm{MP}}$ (the location of the posterior mode). This is equivalent to a standard optimisation of a regularised logistic model, and we use the efficient iteratively-reweighted least-squares algorithm [5] to find the maximum.

2. We compute the Hessian at $\mathbf{w}_{\mathrm{MP}}$:

$$\nabla\nabla \log p(\mathbf{t}, \mathbf{w}|\boldsymbol{\alpha})\big|_{\mathbf{w}_{\mathrm{MP}}} = -(\boldsymbol{\Phi}^{\mathrm{T}}\mathbf{B}\boldsymbol{\Phi} + \mathbf{A}), \qquad (12)$$

    where $\mathbf{B}_{nn} = \sigma\{y(\mathbf{x}_n)\}\left[1 - \sigma\{y(\mathbf{x}_n)\}\right]$, and this is negated and inverted to give the covariance $\boldsymbol{\Sigma}$ for a Gaussian approximation to the posterior over weights, and from that the hyperparameters $\boldsymbol{\alpha}$ are updated using (9). Note that there is no 'noise' variance $\sigma^2$ here.

This procedure is repeated until some suitable convergence criteria are satisfied. Note that in the Bayesian treatment of multilayer neural networks, the Gaussian approximation is considered a weakness of the method if the posterior mode is unrepresentative of the overall probability mass. However, for the RVM, we note that $p(\mathbf{t}, \mathbf{w}|\boldsymbol{\alpha})$ is log-concave (i.e. the Hessian is negative-definite everywhere), which gives us considerably more confidence in the Gaussian approximation.

## 5   Examples of RVM Classification

### 5.1   Synthetic example: Gaussian mixture data

We first utilise artificially generated data in two dimensions in order to illustrate graphically the selection of relevance vectors. Class 1 (denoted by '×') was sampled from a single Gaussian, and overlaps to a small degree class 2 ('•'), sampled from a mixture of two Gaussians.

A relevance vector classifier was compared to its support vector counterpart, using the same Gaussian kernel. A value of $C$ for the SVM was selected using 5-fold cross-validation on the training set. The results for a typical dataset of 200 examples are given in Figure 2. The test errors for the RVM (9.32%) and SVM (9.48%) are comparable, but the remarkable feature of contrast is the complexity of the classifiers. The support vector machine utilises 44 kernel functions compared to just 3 for the relevance vector method.

It is also notable that the relevance vectors are some distance from the decision boundary (in $\mathbf{x}$-space). Given further analysis, this observation can be seen to be consistent with the hyperparameter update equations. A more qualitative explanation is that the output of a basis function lying on or near the decision boundary is a poor indicator of class membership, and such basis functions are naturally 'penalised' under the Bayesian framework.

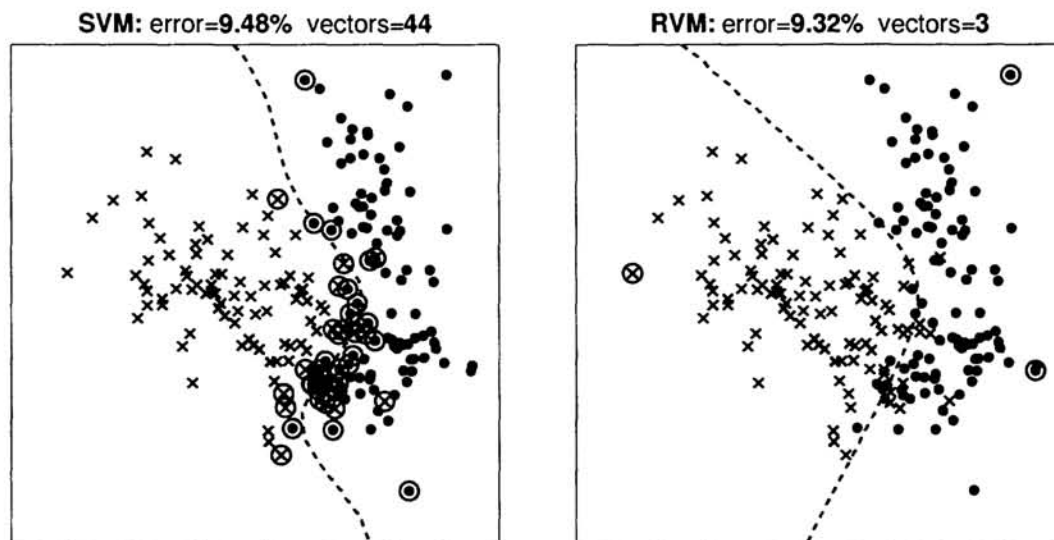

Figure 2: Results of training functionally identical SVM (left) and RVM (right) classifiers on a typical synthetic dataset. The decision boundary is shown dashed, and relevance/support vectors are shown circled to emphasise the dramatic reduction in complexity of the RVM model.

## 5.2 Real examples

In the table below we give error and complexity results for the 'Pima Indian diabetes' and the 'U.S.P.S. handwritten digit' datasets. The former task has been recently used to illustrate Bayesian classification with the related *Gaussian Process* (GP) technique [9], and we utilised those authors' split of the data into 200 training and 332 test examples and quote their result for the GP case. The latter dataset is a popular support vector benchmark, comprising 7291 training examples along with a 2007-example test set, and the SVM result is quoted from [7].

| Dataset | errors | | | kernels | | |
|---|---|---|---|---|---|---|
| | SVM | GP | RVM | SVM | GP | RVM |
| Pima Indians | 67 | 68 | 65 | 109 | 200 | 4 |
| U.S.P.S. | 4.4% | – | 5.1% | 2540 | – | 316 |

In terms of prediction accuracy, the RVM is marginally superior on the Pima set, but outperformed by the SVM on the digit data. However, consistent with other examples in this paper, the RVM classifiers utilise many fewer kernel functions. Most strikingly, the RVM achieves state-of-the-art performance on the diabetes dataset with only 4 kernels. It should be noted that *reduced set* methods exist for subsequently pruning support vector models to reduce the required number of kernels at the expense of some increase in error (*e.g.* see [7] for some example results on the U.S.P.S. data).

## 6 Discussion

Examples in this paper have effectively demonstrated that the relevance vector machine can attain a comparable (and for regression, apparently superior) level of generalisation accuracy as the well-established support vector approach, while at the same time utilising dramatically fewer kernel functions — implying a considerable

saving in memory and computation in a practical implementation. Importantly, we also benefit from the absence of any additional nuisance parameters to set, apart from the need to choose the type of kernel and any associated parameters.

In fact, for the case of kernel parameters, we have obtained improved (both in terms of accuracy and sparsity) results for all the benchmarks given in Section 3.2 when optimising the marginal likelihood with respect to multiple input scale parameters in Gaussian kernels ($q.v.$ [9]). Furthermore, we may also exploit the Bayesian formalism to guide the choice of kernel itself [2], and it should be noted that the presented methodology is applicable to arbitrary basis functions, so we are not limited, for example, to the use of 'Mercer' kernels as in the SVM.

A further advantage of the RVM classifier is its standard formulation as a probabilistic generalised linear model. This implies that it can be extended to the multiple-class case in a straightforward and principled manner, without the need to train and heuristically combine multiple dichotomous classifiers as is standard practice for the SVM. Furthermore, the estimation of posterior probabilities of class membership is a major benefit, as these convey a principled measure of uncertainty of prediction, and are essential if we wish to allow adaptation for varying class priors, along with incorporation of asymmetric misclassification costs.

However, it must be noted that the principal disadvantage of relevance vector methods is in the complexity of the training phase, as it is necessary to repeatedly compute and invert the Hessian matrix, requiring $O(N^2)$ storage and $O(N^3)$ computation. For large datasets, this makes training considerably slower than for the SVM. Currently, memory constraints limit us to training on no more than 5,000 examples, but we have developed approximation methods for handling larger datasets which were employed on the U.S.P.S. handwritten digit database. We note that while the case for Bayesian methods is generally strongest when data is scarce, the sparseness of the resulting classifier induced by the Bayesian framework presented here is a compelling motivation to apply relevance vector techniques to larger datasets.

### Acknowledgements

The author wishes to thank Chris Bishop, John Platt and Bernhard Schölkopf for helpful discussions, and JP again for his Sequential Minimal Optimisation code.

# References

[1] J. O. Berger. *Statistical decision theory and Bayesian analysis.* Springer, New York, second edition, 1985.

[2] D. J. C. Mackay. Bayesian interpolation. *Neural Computation*, 4(3):415–447, 1992.

[3] D. J. C. Mackay. The evidence framework applied to classification networks. *Neural Computation*, 4(5):720–736, 1992.

[4] D. J. C. Mackay. Bayesian non-linear modelling for the prediction competition. In *ASHRAE Transactions*, vol. 100, pages 1053–1062. ASHRAE, Atlanta, Georgia, 1994.

[5] I. T. Nabney. Efficient training of RBF networks for classification. In *Proceedings of ICANN99*, pages 210–215, London, 1999. IEE.

[6] R. M. Neal. *Bayesian Learning for Neural Networks.* Springer, New York, 1996.

[7] B. Schölkopf, S. Mika, C. J. C. Burges, P. Knirsch, K.-R. Müller, G. Rätsch, and A. J. Smola. Input space versus feature space in kernel-based methods. *IEEE Transactions on Neural Networks*, 10(5):1000–1017, 1999.

[8] V. N. Vapnik. *Statistical Learning Theory.* Wiley, New York, 1998.

[9] C. K. I. Williams and D. Barber. Bayesian classification with Gaussian processes. *IEEE Trans. Pattern Analysis and Machine Intelligence*, 20(12):1342–1351, 1998.
